# Blind one-microphone speech separation: A spectral learning approach

**Francis R. Bach**
Computer Science
University of California
Berkeley, CA 94720
*fbach@cs.berkeley.edu*

**Michael I. Jordan**
Computer Science and Statistics
University of California
Berkeley, CA 94720
*jordan@cs.berkeley.edu*

## Abstract

We present an algorithm to perform blind, one-microphone speech separation. Our algorithm separates mixtures of speech without modeling individual speakers. Instead, we formulate the problem of speech separation as a problem in segmenting the spectrogram of the signal into two or more disjoint sets. We build feature sets for our segmenter using classical cues from speech psychophysics. We then combine these features into parameterized affinity matrices. We also take advantage of the fact that we can generate training examples for segmentation by artificially superposing separately-recorded signals. Thus the parameters of the affinity matrices can be tuned using recent work on learning spectral clustering [1]. This yields an adaptive, speech-specific segmentation algorithm that can successfully separate one-microphone speech mixtures.

## 1 Introduction

The problem of recovering signals from linear mixtures, with only partial knowledge of the mixing process and the signals—a problem often referred to as *blind source separation*—is a central problem in signal processing. It has applications in many fields, including speech processing, network tomography and biomedical imaging [2]. When the problem is over-determined, i.e., when there are no more signals to estimate (the sources) than signals that are observed (the sensors), generic assumptions such as statistical independence of the sources can be used in order to demix successfully [2]. Many interesting applications, however, involve under-determined problems (more sources than sensors), where more specific assumptions must be made in order to demix. In problems involving at least two sensors, progress has been made by appealing to sparsity assumptions [3, 4].

However, the most extreme case, in which there is only one sensor and two or more sources, is a much harder and still-open problem for complex signals such as speech. In this setting, simple generic statistical assumptions do not suffice. One approach to the problem involves a return to the spirit of classical engineering methods such as matched filters, and estimating specific models for specific sources—e.g., specific speakers in the case of speech [5, 6]. While such an approach is reasonable, it departs significantly from the desideratum of "blindness." In this paper we present an algorithm that is a blind separation algorithm—our algorithm separates speech mixtures from a single microphone without requiring models of specific speakers.

Our approach involves a "discriminative" approach to the problem of speech separation. That is, rather than building a complex model of speech, we instead focus directly on the task of separation and optimize parameters that determine separation performance. We work within a time-frequency representation (a spectrogram), and exploit the sparsity of speech signals in this representation. That is, although two speakers might speak simultaneously, there is relatively little overlap in the time-frequency plane if the speakers are different [5, 4]. We thus formulate speech separation as a problem in segmentation in the time-frequency plane. In principle, we could appeal to classical segmentation methods from vision (see, e.g. [7]) to solve this two-dimensional segmentation problem. Speech segments are, however, very different from visual segments, reflecting very different underlying physics. Thus we must design features for segmenting speech from first principles.

It also proves essential to combine knowledge-based feature design with learning methods. In particular, we exploit the fact that in speech we can generate "training examples" by artificially superposing two separately-recorded signals. Making use of our earlier work on learning methods for spectral clustering [1], we use the training data to optimize the parameters of a spectral clustering algorithm. This yields an adaptive, "discriminative" segmentation algorithm that is optimized to separate speech signals.

We highlight one other aspect of the problem here—the major computational challenge involved in applying spectral methods to speech separation. Indeed, four seconds of speech sampled at 5.5 KHz yields 22,000 samples and thus we need to manipulate affinity matrices of dimension at least $22,000 \times 22,000$. Thus a major part of our effort has involved the design of numerical approximation schemes that exploit the different time scales present in speech signals.

The paper is structured as follows. Section 2 provides a review of basic methodology. In Section 3 we describe our approach to feature design based on known cues for speech separation [8, 9]. Section 4 shows how parameterized affinity matrices based on these cues can be optimized in the spectral clustering setting. We describe our experimental results in Section 5 and present our conclusions in Section 6.

## 2 Speech separation as spectrogram segmentation

In this section, we first review the relevant properties of speech signals in the time-frequency representation and describe how our training sets are constructed.

### 2.1 Spectrogram

The spectrogram is a two-dimensional (time and frequency) redundant representation of a one-dimensional signal [10]. Let $f[t], t = 0, \ldots, T - 1$ be a signal in $\mathbb{R}^T$. The spectrogram is defined through windowed Fourier transforms and is commonly referred to as a short-time Fourier transform or as Gabor analysis [10]. The value $(Uf)_{mn}$ of the spectrogram at time window $n$ and frequency $m$ is defined as $(Uf)_{mn} = \frac{1}{\sqrt{M}} \sum_{t=0}^{T-1} f[t]w[t - na]e^{i2\pi mt/M}$, where $w$ is a window of length $T$ with small support of length $c$. We assume that the number of samples $T$ is an integer multiple of $a$ and $c$. There are then $N = T/a$ different windows of length $c$. The spectrogram is thus an $N \times M$ image which provides a redundant time-frequency representation of time signals[1] (see Figure 1).

**Inversion** Our speech separation framework is based on the segmentation of the spectrogram of a signal $f[t]$ in $S \geqslant 2$ disjoint subsets $A_i, i = 1, \ldots, S$ of $[0, N - 1] \times [0, M - 1]$.

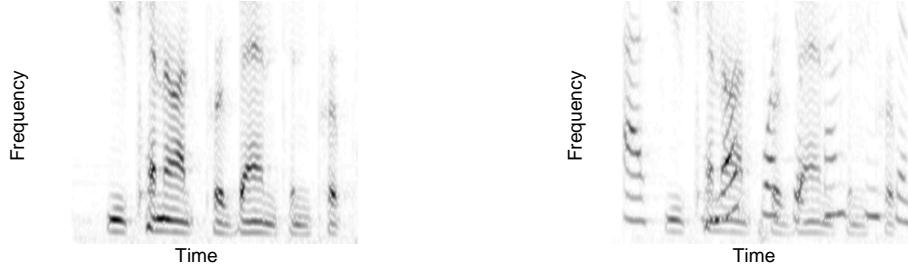

Figure 1: Spectrogram of speech; (left) single speaker, (right) two simultaneous speakers. The gray intensity is proportional to the magnitude of the spectrogram.

This leads to $S$ spectrograms $U_i$ such that $(U_i)_{mn} = U_{mn}$ if $(m, n) \in A_i$ and zero otherwise—note that the phase is kept the same as the one of the original mixed signal. We now need to find $S$ speech signals $f_i[t]$ such that each $U_i$ is the spectrogram of $f_i$. In general there are no exact solutions (because the representation is redundant), and a classical technique is to find the minimum $L_2$ norm approximation, i.e., find $f_i$ such that $||U_i - Uf_i||^2$ is minimal [10]. The solution of this minimization problem involves the pseudo-inverse of the linear operator $U$ [10] and is equal to $f_i = (U^*U)^{-1}U^*U_i$. By our choice of window (Hanning), $U^*U$ is proportional to the identity matrix, so that the solution to this problem can simply be obtained by applying the adjoint operator $U^*$.

**Normalization and subsampling**    There are several ways of normalizing a speech signal. In this paper, we chose to rescale all speech signals as follows: for each time window $n$, we compute the total energy $e_n = \sum_m |Uf_{mn}|^2$, and its 20-point moving average. The signals are normalized so that the $80\%$ percentile of those values is equal to one.

In order to reduce the number of spectrogram samples to consider, for a given pre-normalized speech signal, we threshold coefficients whose magnitudes are less than a value that was chosen so that the distortion is inaudible.

## 2.2   Generating training samples

Our approach is based on a learning algorithm that optimizes a segmentation criterion. The training examples that we provide to this algorithm are obtained by mixing separately-normalized speech signals. That is, given two volume-normalized speech signals $f_1$, $f_2$ of the same duration, with spectrograms $U_1$ and $U_2$, we build a training sample as $U^{train} = U_1 + U_2$, with a segmentation given by $z = \arg\min\{U_1, U_2\}$. In order to obtain better training partitions (and in particular to be more robust to the choice of normalization), we also search over all $\alpha \in [0, 1]$ such that the least square reconstruction error of the waveform obtained from segmenting/reconstructing using $z = \arg\min\{\alpha U_1, (1 - \alpha)U_2\}$ is minimized. An example of such a partition is shown in Figure 2 (left).

## 3   Features and grouping cues for speech separation

In this section we describe our approach to the design of features for the spectral segmentation. We base our design on classical cues suggested from studies of perceptual grouping [11]. Our basic representation is a "feature map," a two-dimensional representation that has the same layout as the spectrogram. Each of these cues is associated with a specific time scale, which we refer to as "small" (less than 5 frames), "medium" (10 to 20 frames), and "large" (across all frames). (These scales will be of particular relevance to the design of numerical approximation methods in Section 4.3). Any given feature is not sufficient for separating by itself; rather, it is the combination of several features that makes our approach successful.

### 3.1 Non-harmonic cues

The following non-harmonic cues have counterparts in visual scenes and for these cues we are able to borrow from feature design techniques used in image segmentation [7].

**Continuity**   Two time-frequency points are likely to belong to the same segment if they are close in time or frequency; we thus use time and frequency directly as features. This cue acts at a small time scale.

**Common fate cues**   Elements that exhibit the same time variation are likely to belong to the same source. This takes several particular forms. The first is simply *common offset* and *common onset*. We thus build an offset map and an onset map, with elements that are zero when no variation occurs, and are large when there is a sharp decrease or increase (with respect to time) for that particular time-frequency point. The onset and offset maps are built using oriented energy filters as used in vision (with one vertical orientation). These are obtained by convolving the spectrogram with derivatives of Gaussian windows [7].

Another form of the common fate cue is *frequency co-modulation*, the situation in which frequency components of a single source tend to move in sync. To capture this cue we simply use oriented filter outputs for a set of orientation angles (8 in our simulations). Those features act mainly at a medium time scale.

### 3.2 Harmonic cues

This is the major cue for voiced speech [12, 9, 8], and it acts at all time scales (small, medium and large): voiced speech is locally periodic and the local period is usually referred to as the pitch.

**Pitch estimation**   In order to use harmonic information, we need to estimate potentially several pitches. We have developed a simple pattern matching framework for doing this that we present in Appendix A. If $S$ pitches are sought, the output that we obtain from the pitch extractor is, for each time frame $n$, the $S$ pitches $\omega_{n1}, \ldots, \omega_{nS}$, as well as the strength $y_{nms}$ of the $s$-th pitch for each frequency $m$.

**Timbre**   The pitch extraction algorithm presented in Appendix A also outputs the spectral envelope of the signal [12]. This can be used to design an additional feature related to timbre which helps integrate information regarding speaker identification across time. Timbre can be loosely defined as the set of properties of a voiced speech signal once the pitch has been factored out [8]. We add the spectral envelope as a feature (reducing its dimensionality using principal component analysis).

**Building feature maps from pitch information**   We build a set of features from the pitch information. Given a time-frequency point $(m, n)$, let $s(m,n) = \arg\max_s \frac{y_{nms}}{(\sum_{m'} y_{nm's})^{1/2}}$ denote the highest energy pitch, and define the features $\omega_{ns(m,n)}$, $y_{nms(m,n)}$, $\sum_{m'} y_{nm's(m,n)}$, $\frac{y_{nms(m,n)}}{\sum_{m'} y_{nm's(m,n)}}$ and $\frac{y_{nms(m,n)}}{(\sum_{m'} y_{nm's(m,n)})^{1/2}}$. We use a partial normalization with the square root to avoid including very low energy signals, while allowing a significant difference between the local amplitude of the speakers.

Those features all come with some form of energy level and all features involving pitch values $\omega$ should take this energy into account when the affinity matrix is built in Section 4. Indeed, the values of the harmonic features have no meaning when no energy in that pitch is present.

## 4   Spectral clustering and affinity matrices

Given the features described in the previous section, we now show how to build affinity (i.e., similarity) matrices that can be used to define a spectral segmenter. In particular, our

approach builds *parameterized* affinity matrices, and uses a learning algorithm to adjust these parameters.

## 4.1 Spectral clustering

Given $P$ data points to partition into $S \geqslant 2$ disjoint groups, spectral clustering methods use an *affinity matrix* $W$, symmetric of size $P \times P$, that encodes topological knowledge about the problem. Once $W$ is available, it is normalized and its first $S$ ($P$-dimensional) eigenvectors are computed. Then, forming a $P \times S$ matrix with these eigenvectors as columns, we cluster the $P$ rows of this matrix as points in $\mathbb{R}^S$ using $K$-means (or a weighted version thereof). These clusters define the final partition [7, 1].

We prefer spectral clustering methods over other clustering algorithms such as $K$-means or mixtures of Gaussians estimated by the EM algorithm because we do not have any reason to expect the segments of interest in our problem to form convex shapes in the feature representation.

## 4.2 Parameterized affinity matrices

The success of spectral methods for clustering depends heavily on the construction of the affinity matrix $W$. In [1], we have shown how learning can play a role in optimizing over affinity matrices. Our algorithm assumes that fully partitioned datasets are available, and uses these datasets as training data for optimizing the parameters of affinity matrices. As we have discussed in Section 2.2, such training data are easily obtained in the speech separation setting. It remains for us to describe how we parameterize the affinity matrices.

From each of the features defined in Section 3, we define a basis affinity matrix $W_j = W_j(\beta_j)$, where $\beta_j$ is a (vector) parameter. We restrict ourselves to affinity matrices whose elements are between zero and one, and with unit diagonal. We distinguish between harmonic and non-harmonic features. For non-harmonic features, we use a radial basis function to define affinities. Thus, if $f_a$ is the value of the feature for data point $a$, we use a basis affinity matrix defined as $W_{ab} = \exp(-||f_a - f_b||^\beta)$, where $\beta > 1$.

For an harmonic feature, on the other hand, we need to take into account the strength of the feature: if $f_a$ is the value of the feature for data point $a$, with strength $y_a$, we use $W_{ab} = \exp(-|g(y_a, y_b) + \beta_3|^{\beta_4} ||f_a - f_b||^{\beta_2})$, where $g(u, v) = (ue^{\beta_5 u} + ve^{\beta_5 v})/(e^{\beta_5 u} + e^{\beta_5 v})$ ranges from the minimum of $u$ and $v$ for $\beta_5 = -\infty$ to their maximum for $\beta_5 = +\infty$.

Given $m$ basis matrices, we use the following parameterization of $W$: $W = \sum_{k=1}^{K} \gamma_k W_1^{\alpha_{k1}} \times \cdots \times W_m^{\alpha_{km}}$, where the products are taken pointwise. Intuitively, if we consider the values of affinity as soft boolean variables, taking the product of two affinity matrices is equivalent to considering the conjunction of two matrices, while taking the sum can be seen as their disjunction: our final affinity matrix can thus be seen as a disjunctive normal form. For our application to speech separation, we consider a sum of $K = 3$ matrices, one matrix for each time scale. This has the advantage of allowing different approximation schemes for each of the time scales, an issue we address in the following section.

## 4.3 Approximations of affinity matrices

The affinity matrices that we consider are huge, of size at least 50,000 by 50,000. Thus a significant part of our effort has involved finding computationally efficient approximations of affinity matrices.

Let us assume that the time-frequency plane is vectorized by stacking one time frame after the other. In this representation, the time scale of a basis affinity matrix $W$ exerts an effect on the degree of "bandedness" of $W$. The matrix $W$ is said *band-diagonal* with bandwidth

$B$, if for all $i, j, |i - j| \geqslant B \Rightarrow W_{ij} = 0$. On a small time scale, $W$ has a small bandwidth; for a medium time scale, the band is larger but still small compared to the total size of the matrix, while for large scale effects, the matrix $W$ has no band structure. Note that the bandwidth $B$ can be controlled by the coefficient of the radial basis function involving the time feature $n$.

For each of these three cases, we have designed a particular way of approximating the matrix, while ensuring that in each case the time and space requirements are *linear* in the number of time frames.

**Small scale**    If the bandwidth $B$ is very small, we use a simple direct sparse approximation. The complexity of such an approximation grows linearly in the number of time frames.

**Medium and large scale**    We use a low-rank approximation of the matrix $W$ similar in spirit to the algorithm of [13]. If we assume that the index set $\{1, \ldots, P\}$ is partitioned randomly into $I$ and $J$, and that $A = W(I, I)$ and $B = W(J, I)$, then $W(J, I) = B^\top$ (by symmetry) and we approximate $C = W(J, J)$ by a linear combination of the columns in $I$, i.e., $\widehat{C} = BE$, where $E \in \mathbb{R}^{|I| \times |J|}$. The matrix $E$ is chosen so that when the linear combination defined by $E$ is applied to the columns in $I$, the error is minimal, which leads to an approximation of $W(J, J)$ by $B(A^2 + \lambda I)^{-1} AB^\top$.

If $G$ is the dimension of $J$, then the complexity of finding the approximation is $O(G^3 + G^2 P)$, and the complexity of a matrix-vector product with the low-rank approximation is $O(G^2 P)$. The storage requirement is $O(GP)$. For large bandwidths, we use a constant $G$, i.e., we make the assumption that the rank that is required to encode a speaker is independent of the duration of the signals.

For mid-range interactions, we need an approximation whose rank grows with time, but whose complexity does not grow quadratically with time. This is done by using the banded structure of $A$ and $W$. If $\rho$ is the proportion of retained indices, then the complexity of storage and matrix-vector multiplication is $O(P\rho^3 B)$.

## 5    Experiments

We have trained our segmenter using data from four different speakers, with speech signals of duration 3 seconds. There were 28 parameters to estimate using our spectral learning algorithm. For testing, we have use mixes from five speakers that were different from those in the training set.

In Figure 2, for two speakers from the testing set, we show on the left part an example of the segmentation that is obtained when the two speech signals are known in advance (obtained as described in Section 2.2), and on the right side, the segmentation that is output by our algorithm. Although some components of the "black" speaker are missing, the segmentation performance is good enough to obtain audible signals of reasonable quality. The speech samples for this example can de downloaded from `www.cs.berkeley.edu/` `~fbach/speech/` . On this web site, there are additional examples of speech separation, with various speakers, in French and in English.

An important point is that our method does not require to know the speaker in advance in order to demix successfully; rather, it just requires that the two speakers have distinct and far enough pitches most of the time (another but less crucial condition is that one pitch is not too close to twice the other one).

As mentioned earlier, there was a major computational challenge in applying spectral methods to single microphone speech separation. Using the techniques described in Section 4.3, the separation algorithm has linear running time complexity and memory requirement and,

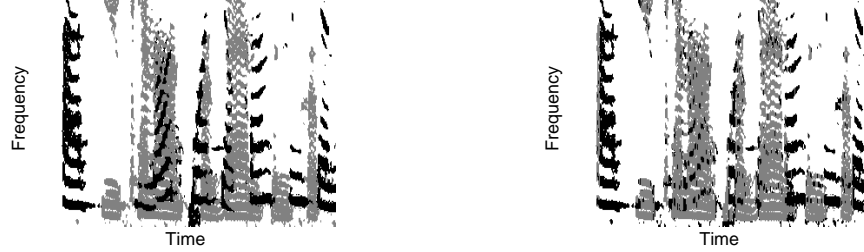

Figure 2: (Left) Optimal segmentation for the spectrogram in Figure 1 (right), where the two speakers are "black" and "grey;" this segmentation is obtained from the known separated signals. (Right) The blind segmentation obtained with our algorithm.

coded in Matlab and C, it takes 30 minutes to separate 4 seconds of speech on a 1.8 GHz processor with 1GB of RAM.

## 6   Conclusions

We have presented an algorithm to perform blind source separation of speech signals from a single microphone. To do so, we have combined knowledge of physical and psychophysical properties of speech with learning methods. The former provide parameterized affinity matrices for spectral clustering, and the latter make use of our ability to generate segmented training data. The result is an optimized segmenter for spectrograms of speech mixtures. We have successfully demixed speech signals from two speakers using this approach.

Our work thus far has been limited to the setting of ideal acoustics and equal-strength mixing of two speakers. There are several obvious extensions that warrant investigation. First, the mixing conditions should be weakened and should allow some form of delay or echo. Second, there are multiple applications where speech has to be separated from a non-stationary noise; we believe that our method can be extended to this situation. Third, our framework is based on segmentation of the spectrogram and, as such, distortions are inevitable since this is a "lossy" formulation [6, 4]. We are currently working on post-processing methods that remove some of those distortions. Finally, while running time and memory requirements of our algorithm are linear in the duration of the signal to be separated, the resource requirements remain a concern. We are currently working on further numerical techniques that we believe will bring our method significantly closer to real-time.

## Appendix A. Pitch estimation

**Pitch estimation for one pitch**    In this paragraph, we assume that we are given one time slice $s$ of the spectrogram magnitude, $s \in \mathbb{R}^M$. The goal is to have a specific pattern match $s$. Since the speech signals are real, the spectrogram is symmetric and we can consider only $M/2$ samples.

If the signal is exactly periodic, then the spectrogram magnitude for that time frame is exactly a superposition of bumps at multiples of the fundamental frequency, The patterns we are considering have thus the following parameters: a "bump" function $u \mapsto b(u)$, a pitch $\omega \in [0, M/2]$ and a sequence of harmonics $x_1, \ldots, x_H$ at frequencies $\omega_1 = \omega, \ldots, \omega_H = H\omega$, where $H$ is the largest acceptable harmonic multiple, i.e., $H = \lfloor M/2\omega \rfloor$. The pattern $\tilde{s} = \tilde{s}(x, b, \omega)$ is then built as a weighted sum of bumps.

By pattern matching, we mean to find the pattern $\tilde{s}$ as close to $s$ in the $L^2$-norm sense. We impose a constraint on the harmonic strengths $(x_h)$, namely, that they are samples at $h\omega$ of a function $g$ with small second derivative norm $\int_0^{M/2} |g^{(2)}(\omega)|^2 d\omega$. The function $g$ can

be seen as the envelope of the signal and is related to the "timbre" of the speaker [8]. The explicit consideration of the envelope and its smoothness is necessary for two reasons: (a) it will provide a timbre feature helpful for separation, (b) it helps avoid pitch-halving, a traditional problem of pitch extractors [12].

Given $b$ and $\omega$, we minimize with respect to $x$, $||s - \tilde{s}(x)||^2 + \lambda \int_0^{M/2} |g^{(2)}(\omega)|^2 d\omega$, where $x_h = g(h\omega)$. Since $\tilde{s}(x)$ is linear function of $x$, this is a spline smoothing problem, and the solution can be obtained in closed form with complexity $O(H^3)$ [14].

We now have to search over $b$ and $\omega$, knowing that the harmonic strengths $x$ can be found in closed form. We use exhaustive search on a grid for $\omega$, while we take only a few bump shapes. The main reason for several bump shapes is to account for the only approximate periodicity of voiced speech. For further details and extensions, see [15].

**Pitch estimation for several pitches** If we are to estimate $S$ pitches, we estimate them recursively, by removing the estimated harmonic signals. In this paper, we assume that the number of speakers and hence the maximum number of pitches is known. Note, however, that since all our pitch features are always used with their strengths, our separation method is relatively robust to situations where we try to look for too many pitches.

### Acknowledgments

We wish to acknowledge support from a grant from Intel Corporation, and a graduate fellowship to Francis Bach from Microsoft Research.

## Footnotes

[1]In our simulations, the sampling frequency is $f_0 = 5.5 kHz$ and we use a Hanning window of length $c = 216$ (i.e., $43.2ms$). The spacing between window is equal to $a = 54$ (i.e., $10.8ms$). We use a 512-point FFT ($M = 512$). For a speech sample of length 4 sec, we have $T = 22,000$ samples and then $N = 407$, which makes $\approx 2 \times 10^5$ spectrogram pixels.

### References

[1] F. R. Bach and M. I. Jordan. Learning spectral clustering. In *NIPS 16*, 2004.

[2] A. Hyvärinen, J. Karhunen, and E. Oja. *Independent Component Analysis*. John Wiley & Sons, 2001.

[3] M. Zibulevsky, P. Kisilev, Y. Y. Zeevi, and B. A. Pearlmutter. Blind source separation via multinode sparse representation. In *NIPS 14*, 2002.

[4] O. Yilmaz and S. Rickard. Blind separation of speech mixtures via time-frequency masking. *IEEE Trans. Sig. Proc.*, 52(7):1830–1847, 2004.

[5] S. T. Roweis. One microphone source separation. In *NIPS 13*, 2001.

[6] G.-J. Jang and T.-W. Lee. A probabilistic approach to single channel source separation. In *NIPS 15*, 2003.

[7] J. Shi and J. Malik. Normalized cuts and image segmentation. *IEEE PAMI*, 22(8):888–905, 2000.

[8] A. S. Bregman. *Auditory Scene Analysis: The Perceptual Organization of Sound*. MIT Press, 1990.

[9] G. J. Brown and M. P. Cooke. Computational auditory scene analysis. *Computer Speech and Language*, 8:297–333, 1994.

[10] S. Mallat. *A Wavelet Tour of Signal Processing*. Academic Press, 1998.

[11] M. Cooke and D. P. W. Ellis. The auditory organization of speech and other sources in listeners and computational models. *Speech Communication*, 35(3-4):141–177, 2001.

[12] B. Gold and N. Morgan. *Speech and Audio Signal Processing: Processing and Perception of Speech and Music*. Wiley Press, 1999.

[13] S. Belongie, C. Fowlkes, F. Chung, and J. Malik. Spectral partitioning with indefinite kernels using the Nyström extension. In *ECCV*, 2002.

[14] G. Wahba. *Spline Models for Observational Data*. SIAM, 1990.

[15] F. R. Bach and M. I. Jordan. Discriminative training of hidden Markov models for multiple pitch tracking. In *ICASSP*, 2005.
